# Thresholding Procedures for High Dimensional Variable Selection and Statistical Estimation

**Shuheng Zhou**

Seminar für Statistik

ETH Zürich

CH-8092, Switzerland

## Abstract

Given $n$ noisy samples with $p$ dimensions, where $n \ll p$, we show that the multi-step thresholding procedure can accurately estimate a sparse vector $\beta \in \mathbb{R}^p$ in a linear model, under the restricted eigenvalue conditions (Bickel-Ritov-Tsybakov 09). Thus our conditions for model selection consistency are considerably weaker than what has been achieved in previous works. More importantly, this method allows very significant values of $s$, which is the number of non-zero elements in the true parameter. For example, it works for cases where the ordinary Lasso would have failed. Finally, we show that if $X$ obeys a uniform uncertainty principle and if the true parameter is sufficiently sparse, the Gauss-Dantzig selector (Candès-Tao 07) achieves the $\ell_2$ loss within a logarithmic factor of the ideal mean square error one would achieve with an oracle which would supply perfect information about which coordinates are non-zero and which are above the noise level, while selecting a sufficiently sparse model.

## 1   Introduction

In a typical high dimensional setting, the number of variables $p$ is much larger than the number of observations $n$. This challenging setting appears in linear regression, signal recovery, covariance selection in graphical modeling, and sparse approximations. In this paper, we consider recovering $\beta \in \mathbb{R}^p$ in the following linear model:

$$Y = X\beta + \epsilon, \tag{1.1}$$

where $X$ is an $n \times p$ design matrix, $Y$ is a vector of noisy observations and $\epsilon$ is the noise term. We assume throughout this paper that $p \geq n$ (i.e. high-dimensional), $\epsilon \sim N(0, \sigma^2 I_n)$, and the columns of $X$ are normalized to have $\ell_2$ norm $\sqrt{n}$. Given such a linear model, two key tasks are to identify the relevant set of variables and to estimate $\beta$ with bounded $\ell_2$ loss.

In particular, recovery of the sparsity pattern $S = \text{supp}(\beta) := \{j : \beta_j \neq 0\}$, also known as variable (model) selection, refers to the task of correctly identifying the support set (or a subset of "significant" coefficients in $\beta$) based on the noisy observations. Even in the noiseless case, recovering $\beta$ (or its support) from $(X, Y)$ seems impossible when $n \ll p$. However, a line of recent research shows that it becomes possible when $\beta$ is also sparse: when it has a relatively small number of nonzero coefficients and when the design matrix $X$ is also sufficiently nice, which we elaborate below. One important stream of research, which we also adopt here, requires computational feasibility for the estimation methods, among which the Lasso and the Dantzig selector are both well studied and shown with provable nice statistical properties; see for example [11, 9, 19, 21, 5, 18, 12, 2]. For a chosen penalization parameter $\lambda_n \geq 0$, regularized estimation with the $\ell_1$-norm penalty, also known

as the Lasso [16] or Basis Pursuit [6] refers to the following convex optimization problem

$$\widehat{\beta} = \arg\min_{\beta} \frac{1}{2n} \|Y - X\beta\|_2^2 + \lambda_n \|\beta\|_1, \tag{1.2}$$

where the scaling factor $1/(2n)$ is chosen by convenience; The Dantzig selector [5] is defined as,

$$(DS) \quad \arg\min_{\widehat{\beta} \in \mathbb{R}^p} \left\|\widehat{\beta}\right\|_1 \quad \text{subject to} \quad \left\|\frac{1}{n} X^T (Y - X\widehat{\beta})\right\|_\infty \leq \lambda_n. \tag{1.3}$$

Our goal in this work is to recover $S$ as accurately as possible: we wish to obtain $\widehat{\beta}$ such that $|\operatorname{supp}(\widehat{\beta}) \setminus S|$ (and sometimes $|S \triangle \operatorname{supp}(\widehat{\beta})|$ also) is small, with high probability, while at the same time $\|\widehat{\beta} - \beta\|_2^2$ is bounded within logarithmic factor of the ideal mean square error one would achieve with an oracle which would supply perfect information about which coordinates are non-zero and which are above the noise level (hence achieving the *oracle inequality* as studied in [7, 5]); We deem the bound on $\ell_2$-loss as a natural criteria for evaluating a sparse model when it is not exactly $S$. Let $s = |S|$. Given $T \subseteq \{1, \ldots, p\}$, let us define $X_T$ as the $n \times |T|$ submatrix obtained by extracting columns of $X$ indexed by $T$; similarly, let $\beta_T \in \mathbb{R}^{|T|}$, be a subvector of $\beta \in \mathbb{R}^p$ confined to $T$.

Formally, we study a **Multi-step Procedure**: First we obtain an initial estimator $\beta_{\text{init}}$ using the Lasso as in (1.2) or the Dantzig selector as in (1.3), with $\lambda_n = \Theta(\sigma\sqrt{2\log p/n})$.

1. We then threshold the estimator $\beta_{\text{init}}$ with $t_0$, with the general goal such that, we get a set $I_1$ with cardinality at most $2s$; in general, we also have $|I_1 \cup S| \leq 2s$, where $I_1 = \{j \in \{1, \ldots, p\} : \beta_{j,\text{init}} \geq t_0\}$ for some $t_0$ to be specified. Set $I = I_1$.

2. We then feed $(Y, X_I)$ to either the Lasso estimator as in (1.2) or the ordinary least squares (OLS) estimator to obtain $\widehat{\beta}$, where we set $\widehat{\beta}_I = (X_I^T X_I)^{-1} X_I^T Y$ and $\widehat{\beta}_{I^c} = 0$.

3. We then possibly threshold $\widehat{\beta}_{I_1}$ with $t_1 = 4\lambda_n\sqrt{|I_1|}$ (to be specified), to obtain $I_2$, repeat step 2 with $I = I_2$ to obtain $\widehat{\beta}_I$ and set all other coordinates to zero; return $\widehat{\beta}$.

Our algorithm is constructive in that it does not rely on the unknown parameters $s$, $\beta_{\min} := \min_{j \in S} |\beta_j|$ or those that characterize the incoherence conditions on $X$; instead, our choice of $\lambda_n$ and thresholding parameters only depends on $\sigma, n$, and $p$. In our experiments, we apply only the first two steps, which we refer to as a *two-step procedure*; In particular, the Gauss-Dantzig selector is a two-step procedure with the Dantzig selector as $\beta_{\text{init}}$ [5]. In theory, we apply the third step only when $\beta_{\min}$ is sufficiently large and when we wish to get a "sparser" model $I$.

**More definitions.** For a matrix $A$, let $\Lambda_{\min}(A)$ and $\Lambda_{\max}(A)$ denote the smallest and the largest eigenvalues respectively. We refer to a vector $\upsilon \in \mathbb{R}^p$ with at most $s$ non-zero entries, where $s \leq p$, as a $s$-**sparse** vector. Throughout this paper, we assume that $n \geq 2s$ and

$$\Lambda_{\min}(2s) \overset{\triangle}{=} \min_{\upsilon \neq 0; 2s-\text{sparse}} \|X\upsilon\|_2^2 / (n \|\upsilon\|_2^2) > 0. \tag{1.4}$$

It is clear that $n \geq 2s$ is necessary, as any submatrix with more than $n$ columns must be singular. In general, we also assume $\Lambda_{\max}(s) \overset{\triangle}{=} \max_{\upsilon \neq 0; s-\text{sparse}} \|X\upsilon\|_2^2 / (n \|\upsilon\|_2^2) < \infty$. As defined in [4], the $s$-restricted isometry constant $\delta_s$ of $X$ is the smallest quantity such that

$$(1 - \delta_s) \|\upsilon\|_2^2 \leq \|X_T \upsilon\|_2^2 / n \leq (1 + \delta_s) \|\upsilon\|_2^2,$$

for all $T \subseteq \{1, \ldots, p\}$ with $|T| \leq s$ and coefficients sequences $(\upsilon_j)_{j \in T}$. It is clear that $\delta_s$ is non-decreasing in $s$ and $1 - \delta_s \leq \Lambda_{\min}(s) \leq \Lambda_{\max}(s) \leq 1 + \delta_s$. Hence $\delta_{2s} < 1$ implies (1.4). Occasionally, we use $\beta_T \in \mathbb{R}^{|T|}$, where $T \subseteq \{1, \ldots, p\}$, to also represent its 0-extended version $\beta' \in \mathbb{R}^p$ such that $\beta'_{T^c} = 0$ and $\beta'_T = \beta_T$; for example in (1.5) below.

**Oracle inequalities.** The following idea has been explained in [5]; we hence describe it here only briefly. Note that due to different normalization of columns of $X$, our expressions are slightly

different from those in [5]. Consider the least square estimator $\widehat{\beta}_I = (X_I^T X_I)^{-1} X_I^T Y$, where $|I| \leq s$ and consider the *ideal* least-squares estimator $\beta^\diamond$

$$\beta^\diamond = \operatorname*{arg\,min}_{I \subseteq \{1,\ldots,p\},\ |I| \leq s} \mathbf{E} \left\| \beta - \widehat{\beta}_I \right\|_2^2, \tag{1.5}$$

which minimizes the expected mean squared error. It follows from [5] that for $\Lambda_{\max}(s) < \infty$,

$$\mathbf{E} \left\| \beta - \beta^\diamond \right\|_2^2 \geq \min\left(1, 1/\Lambda_{\max}(s)\right) \sum_{i=1}^p \min(\beta_i^2, \sigma^2/n). \tag{1.6}$$

Now we check if for $\Lambda_{\max}(s) < \infty$, it holds with high probability that

$$\left\| \widehat{\beta} - \beta \right\|_2^2 = O(\log p) \sum_{i=1}^p \min(\beta_i^2, \sigma^2/n), \quad \text{so that} \tag{1.7}$$

$$\left\| \widehat{\beta} - \beta \right\|_2^2 = O(\log p) \max(1, \Lambda_{\max}(s)) \mathbf{E} \left\| \beta^\diamond - \beta \right\|_2^2 \quad \text{in view of (1.6).} \tag{1.8}$$

These bounds are meaningful since

$$\sum_{i=1}^p \min(\beta_i^2, \sigma^2/n) = \min_{I \subseteq \{1,\ldots,p\}} \| \beta - \beta_I \|_2^2 + \frac{|I|\sigma^2}{n},$$

represents the ideal squared bias and variance. We elaborate on conditions on the design, under which we accomplish these goals using the multi-step procedures in the rest of this section. We now define a constant $\lambda_{\sigma,a,p}$ for each $a > 0$, by which we bound the maximum correlation between the noise and covariates of $X$, which we only apply to $X$ with column $\ell_2$ norm bounded by $\sqrt{n}$; Let

$$\mathcal{T}_a := \left\{ \epsilon : \left\| \frac{X^T \epsilon}{n} \right\|_\infty \leq \lambda_{\sigma,a,p} \right\}, \quad \text{where } \lambda_{\sigma,a,p} = \sigma\sqrt{1+a}\sqrt{\frac{2\log p}{n}}, \quad \text{hence} \tag{1.9}$$

$$\mathbb{P}\left(\mathcal{T}_a\right) \geq 1 - (\sqrt{\pi \log p}\, p^a)^{-1}, \quad \text{for } a \geq 0; \text{ see [5]}. \tag{1.10}$$

**Variable selection.** Our first result in Theorem 1.1 shows that consistent variable selection is possible under the Restricted Eigenvalue conditions, as formalized in [2]. Similar conditions have been used by [10] and [17].

**Assumption 1.1** (**Restricted Eigenvalue assumption** $RE(s, k_0, X)$ **[2]**) *For some integer* $1 \leq s \leq p$ *and a positive number* $k_0$, *the following holds:*

$$\frac{1}{K(s, k_0, X)} \overset{\triangle}{=} \min_{\substack{J_0 \subseteq \{1,\ldots,p\}, \\ |J_0| \leq s}} \min_{\substack{v \neq 0, \\ \left\| v_{J_0^c} \right\|_1 \leq k_0 \left\| v_{J_0} \right\|_1}} \frac{\|Xv\|_2}{\sqrt{n}\, \|v_{J_0}\|_2} > 0. \tag{1.11}$$

If $RE(s, k_0, X)$ is satisfied with $k_0 \geq 1$, then the square submatrices of size $\leq 2s$ of $X^T X$ are necessarily positive definite (see [2]) and hence (1.4) must hold. We do not impose any extra constraint on $s$ besides what is allowed in order for (1.11) to hold. Note that when $s > n/2$, it is impossible for the restricted eigenvalue assumption to hold as $X_I$ for any $I$ such that $|I| = 2s$ becomes singular in this case. Hence our algorithm is especially relevant if one would like to estimate a parameter $\beta$ such that $s$ is very close to $n$; See Section 4 for such examples. Let $\beta_{\min} := \min_{j \in S} |\beta_j|$.

**Theorem 1.1** (**Variable selection under Assumption 1.1**) *Suppose that* $RE(s, k_0, X)$ *condition holds, where* $k_0 = 1$ *for the DS and* $= 3$ *for the Lasso. Suppose* $\lambda_n \geq B\lambda_{\sigma,a,p}$ *for* $\lambda_{\sigma,a,p}$ *as in (1.9), where* $B \geq 1$ *for the DS and* $\geq 2$ *for the Lasso. Let* $B_2 = \frac{1}{B\Lambda_{\min}(2s)}$. *Let* $s \geq K^4(s, k_0, X)$ *and*

$$\beta_{\min} \geq 4\sqrt{2} \max(K(s, k_0, X), 1)\lambda_n \sqrt{s} + \max\left(4K^2(s, k_0, X), \sqrt{2}B_2\right)\lambda_n \sqrt{s}.$$

*Then with probability at least* $\mathbb{P}\left(\mathcal{T}_a\right)$, *the multi-step procedure returns* $\widehat{\beta}$ *such that*

$$S \subseteq I := \operatorname{supp}(\widehat{\beta}), \quad \text{where } |I \setminus S| < \frac{B_2^2}{16} \quad \text{and}$$

$$\|\widehat{\beta} - \beta\|_2^2 \leq \frac{\lambda_{\sigma,a,p}^2 |I|}{\Lambda_{\min}^2(|I|)} \leq \frac{2\log p(1+a)s\sigma^2(1 + B_2^2/16)}{n\Lambda_{\min}^2(2s)},$$

*which satisfies (1.7) and (1.8) given that* $\beta_{\min} \geq \sigma/\sqrt{n}$ *and* $\sum_{i=1}^p \min(\beta_i^2, \sigma^2/n) = s\sigma^2/n$.

Our analysis builds upon the rate of convergence bounds for $\beta_{\text{init}}$ derived in [2]. The first implication of this work and also one of the motivations for analyzing the thresholding methods is: under Assumption 1.1, one can obtain consistent variable selection for very significant values of $s$, if only a few extra variables are allowed to be included in the estimator $\widehat{\beta}$. In our simulations, we recover the exact support set $S$ with very high probability using a two-step procedure. Note that we did not optimize the lower bound on $s$ as we focus on cases when the support of $S$ is large.

**Thresholding that achieves the oracle inequalities.** The natural question upon obtaining Theorem 1.1 is: is there a good thresholding rule that enables us to obtain a sufficiently sparse estimator $\widehat{\beta}$ when some components of $\beta_S$ (and hence $\beta_{\min}$) are well below $\sigma/\sqrt{n}$, which also satisfies the oracle inequality as in (1.7)? Before we answer this question, we define $s_0$ as the smallest integer such that

$$\sum_{i=1}^{p} \min(\beta_i^2, \lambda^2 \sigma^2) \le s_0 \lambda^2 \sigma^2, \ \text{where} \ \ \lambda = \sqrt{2 \log p / n}, \tag{1.12}$$

and the $(s, s')$-restricted orthogonality constant [4] $\theta_{s,s'}$ as the smallest quantity such that

$$| \langle X_T c, X_{T'} c' \rangle \, /n | \le \theta_{s,s'} \, \|c\|_2 \, \|c'\|_2 \tag{1.13}$$

holds for all disjoint sets $T, T' \subseteq \{1, \ldots, p\}$ of cardinality $|T| \le s$ and $|T'| < s'$, where $s + s' \le p$. Note that $\theta$ is non-decreasing in $s, s'$ and small values of $\theta_{s,s'}$ indicates that disjoint subsets covariates in $X_T$ and $X_{T'}$ span nearly orthogonal subspaces.

Theorem 1.2 says that under a uniform uncertainty principle (UUP), thresholding of an initial Dantzig selector $\beta_{\text{init}}$, at the level of $\Theta(\sigma \sqrt{2 \log p / n})$ indeed identifies a sparse model $I$ of cardinality at most $2 s_0$ such that the $\ell_2^2$-loss for its corresponding least-squares estimator is indeed bounded within $O(\log p)$ of the ideal mean square error as in (1.5), when $\beta$ is as sparse as required by the Dantzig selector to achieve such an oracle inequality [5]. This is accomplished without any knowledge of the significant coordinates of $\beta$ and not being able to observe parameter values.

**Assumption 1.2 (A Uniform Uncertainly Principle) [5]** *For some integer $1 \le s < n/3$, assume $\delta_{2s} + \theta_{s,2s} < 1$, which implies that $\lambda_{\min}(2s) > \theta_{s,2s}$ given that $1 - \delta_{2s} \le \Lambda_{\min}(2s)$.*

**Theorem 1.2** *Choose $\tau, a > 0$ and set $\lambda_n = \lambda_{p,\tau} \sigma$, where $\lambda_{p,\tau} := (\sqrt{1+a} + \tau^{-1}) \sqrt{2 \log p / n}$, in (1.3). Suppose $\beta$ is $s$-sparse with $\delta_{2s} + \theta_{s,2s} < 1 - \tau$. Let threshold $t_0$ be chosen from the range $(C_1 \lambda_{p,\tau} \sigma, C_4 \lambda_{p,\tau} \sigma]$ for some constants $C_1, C_4$ to be defined. Then with probability at least $1 - (\sqrt{\pi \log p} p^a)^{-1}$, the Gauss-Dantzig selector $\widehat{\beta}$ selects a model $I := \text{supp}(\widehat{\beta})$ such that $|I| \le 2 s_0$,*

$$|I \setminus S| \le s_0 \le s, \ \ and \ \ \|\widehat{\beta} - \beta\|_2^2 \le 2 C_3^2 \log p \left( \sigma^2 / n + \sum_{i=1}^{p} \min(\beta_i^2, \sigma^2 / n) \right), \tag{1.14}$$

*where $C_3$ depends on $a, \tau, \delta_{2s}, \theta_{s,2s}$ and $C_4$; see (3.3).*

Our analysis builds upon [5]. Note that allowing $t_0$ to be chosen from a range (as wide as one would like, with the cost of increasing the constant $C_3$ in (1.14)), saves us from having to estimate $C_1$, which indeed depends on $\delta_{2s}$ and $\theta_{s,2s}$. Assumption 1.2 implies that Assumption 1.1 holds for $k_0 = 1$ with $K(s, k_0, X) = \sqrt{\Lambda_{\min}(2s)}/(\Lambda_{\min}(2s) - \theta_{s,2s}) \le \sqrt{\Lambda_{\min}(2s)}/(1 - \delta_{2s} - \theta_{s,2s})$ (see [2]); It is an open question if we can derive the same result under Assumption 1.1.

**Previous work.** Finally, we briefly review related work in multi-step procedures and the role of sparsity for high-dimensional statistical inference. Before this work, hard thresholding idea has been shown in [5] (via Gauss-Dantzig selector) as a method to correct the bias of the initial Dantzig selector. The empirical success of the Gauss-Dantzig selector in terms of improving the statistical accuracy is strongly evident in their experimental results. Our theoretical analysis on the oracle inequalities, which hold for the Gauss-Dantzig selector under a uniform uncertainty principle, is exactly inspired by their theoretical analysis of the initial Dantzig selector under the same conditions. For the Lasso, [12] has also shown in theoretical analysis that thresholding is effective in obtaining

a two-step estimator $\widehat{\beta}$ that is consistent in its support with $\beta$; however, the choice of threshold level depends on the unknown value $\beta_{\min}$ (which needs to be sufficiently large) and $s$, and their theory does not directly yield (or imply) an algorithm for finding such parameters. Further, as pointed out by [2], a weakening of their condition is still sufficient for Assumption 1.1 to hold.

The sparse recovery problem under arbitrary noise is also well studied, see [3, 15, 14]. Although as argued in [3, 14], the best accuracy under arbitrary noise has essentially been achieved in both work, their bounds are worse than that in [5] (hence the present paper) under the stochastic noise as discussed in the present paper; see more discussions in [5]. Moreover, greedy algorithms in [15, 14] require $s$ to be part of their input, while the iterative algorithms in the present paper do not have such requirement, and hence adapt to the unknown level of sparsity $s$ well. A more general framework on multi-step variable selection was studied by [20]. They control the probability of false positives at the price of false negatives, similar to what we aim for in the present paper. Unfortunately, their analysis is constrained to the case when $s$ is a constant. Finally, under a restricted eigenvalue condition slightly stronger than Assumption 1.1, [22] requires $s = O(\sqrt{n/\log p})$ in order to achieve variable selection consistency using the adaptive Lasso [23] as the second step procedure.

**Organization of the paper.** We prove Theorem 1.1 essentially in Section 2. A thresholding framework for the general setting is described in Section 3, which also sketches the proof of Theorem 1.2. Section 4 briefly discusses the relationship between linear sparsity and random design matrices. Section 5 includes simulation results showing that our two-step procedure is consistent with our theoretical analysis on variable selection.

# 2   Thresholding procedure when $\beta_{\min}$ is large

We use a penalization parameter $\lambda_n = B\lambda_{\sigma,a,p}$ and assume $\beta_{\min} > C\lambda_n\sqrt{s}$ for some constants $B, C$ throughout this section; we first specify the thresholding parameters in this case. We then show in Theorem 2.1 that our algorithm works under any conditions so long as the rate of convergence of the initial estimator obeys the bounds in (2.2). Theorem 1.1 is a corollary of Theorem 2.1 under Assumption 1.1, given the rate of convergence bounds for $\beta_{\text{init}}$ following derivations in [2].

**The Iterative Procedure.** We obtain an initial estimator $\beta_{\text{init}}$ using the Lasso or the Dantzig selector. Let $\widehat{S}_0 = \{j : \beta_{j,\text{init}} > 4\lambda_n\}$, and $\widehat{\beta}^{(0)} := \beta_{\text{init}}$; Iterate through the following steps twice, for $i = 0, 1$: (a) Set $t_i = 4\lambda_n\sqrt{|\widehat{S}_i|}$; (b) Threshold $\widehat{\beta}^{(i)}$ with $t_i$ to obtain $I := \widehat{S}_{i+1}$, where

$$\widehat{S}_{i+1} = \left\{ j \in \widehat{S}_i : \widehat{\beta}_j^{(i)} \geq 4\lambda_n\sqrt{|\widehat{S}_i|} \right\} \text{ and compute } \widehat{\beta}_I^{(i+1)} = (X_I^T X_I)^{-1} X_I^T Y. \tag{2.1}$$

Return the final set of variables in $\widehat{S}_2$ and output $\widehat{\beta}$ such that $\widehat{\beta}_{\widehat{S}_2} = \widehat{\beta}_{\widehat{S}_2}^{(2)}$ and $\widehat{\beta}_j = 0, \forall j \in \widehat{S}_2^c$.

**Theorem 2.1** *Let $\lambda_n \geq B\lambda_{\sigma,a,p}$, where $B \geq 1$ is a constant suitably chosen such that the initial estimator $\beta_{\text{init}}$ satisfies on $\mathcal{T}_a$, for $v_{\text{init}} = \beta_{\text{init}} - \beta$ and some constants $B_0, B_1$,*

$$\|v_{\text{init},S}\|_2 \quad \leq \quad B_0\lambda_n\sqrt{s} \text{ and } \|v_{\text{init},S^c}\|_1 \leq B_1\lambda_n s; \tag{2.2}$$

$$Suppose \quad \beta_{\min} \quad \geq \quad \left(\max\left(\sqrt{B_1}, 2\right) 2\sqrt{2} + \max\left(B_0, \sqrt{2}B_2\right)\right) \lambda_n\sqrt{s}, \tag{2.3}$$

*where $B_2 = 1/(B\Lambda_{\min}(2s))$. Then for $s \geq B_1^2/16$, it holds on $\mathcal{T}_a$ that $|\widehat{S}_i| \leq 2s, \forall i = 1, 2$, and*

$$\|\widehat{\beta}^{(i)} - \beta\|_2 \quad \leq \quad \lambda_{\sigma,a,p}\sqrt{|\widehat{S}_i|}/\Lambda_{\min}(|\widehat{S}_i|) \leq \lambda_n B_2\sqrt{2s}, \forall i = 1, 2, \tag{2.4}$$

*where $\widehat{\beta}^{(i)}$ are the OLS estimators based on $I = \widehat{S}_i$; Finally, the Iterative Procedure includes the correct set of variables in $\widehat{S}_2$ such that $S \subseteq \widehat{S}_2 \subseteq \widehat{S}_1$ and*

$$\left|\widehat{S}_2 \setminus S\right| := \left|\text{supp}(\widehat{\beta}) \setminus S\right| \leq \frac{1}{16B^2\Lambda_{\min}^2(|\widehat{S}_1|)} \leq \frac{B_2^2}{16}. \tag{2.5}$$

**Remark 2.2** *Without the knowledge of $\sigma$, one could use $\widehat{\sigma} \geq \sigma$ in $\lambda_n$; this will put a stronger requirement on $\beta_{\min}$, but all conclusions of Theorem 2.1 hold. We also note that in order to obtain $\widehat{S}_1$ such that $|\widehat{S}_1| \leq 2s$ and $\widehat{S}_1 \supseteq S$, we only need to threshold $\beta_{\text{init}}$ at $t_0 = B_1 \lambda_n$ (see Section 3 and Lemma 3.2 for an example); instead of having to estimate $B_1$, we use $t_0 = \Theta(\lambda_n \sqrt{s})$ to threshold.*

## 3 A thresholding framework for the general setting

In this section, we wish to derive a meaningful criteria for consistency in variable selection, when $\beta_{\min}$ is well below the noise level. Suppose that we are given an initial estimator $\beta_{\text{init}}$ that achieves the rate of convergence bound as in (1.14), which adapts nearly ideally to the uncertainty in the support set $S$ and the "significant" set. We show that although we cannot guarantee the presence of variables indexed by $\{j : |\beta_j| < \sigma\sqrt{2\log p/n}\}$ to be included in the final set $I$ (cf. (3.7)) due to their lack of strength, we wish to include the significant variables from $S$ in $I$ such that the OLS estimator based on $I$ achieves this almost ideal rate of convergence as $\beta_{\text{init}}$ does, even though some variables from $S$ are missing in $I$. Here we pay a price for the missing variables in order to obtain a sparse model $I$. Toward this goal, we analyze the following algorithm under Assumption 1.2.

**The General Two-step Procedure**: Assume $\delta_{2s} + \theta_{s,2s} < 1 - \tau$, where $\tau > 0$;

1. First we obtain an initial estimator $\beta_{\min}$ using the Dantzig selector with $\lambda_{p,\tau} := (\sqrt{1+a} + \tau^{-1})\sqrt{2\log p/n}$, where $\tau, a \geq 0$; we then threshold $\beta_{\text{init}}$ with $t_0$, chosen from the range $(C_1 \lambda_{p,\tau}\sigma, C_4\lambda_{p,\tau}\sigma]$, to obtain a set $I$ of cardinality at most $2s$, (we prove a stronger result in Lemma 3.2), where

$$I := \{j \in \{1,\ldots,p\} : \beta_{j,\text{init}} > t_0\}, \quad \text{for } C_1 \text{ as defined in (3.3)}; \qquad (3.1)$$

2. In the second step, given a set $I$ of cardinality at most $2s$, we run the OLS regression to obtain obtained via (3.1), $\widehat{\beta}_I = (X_I^T X_I)^{-1} X_I^T Y$ and set $\widehat{\beta}_j = 0, \forall j \notin I$.

Theorem 2 in [5] has shown that the Dantzig selector achieves nearly the ideal level of MSE.

**Proposition 3.1** [5] *Let $Y = X\beta + \epsilon$, for $\epsilon$ being i.i.d. $N(0,\sigma^2)$ and $\|X_j\|_2^2 = n$. Choose $\tau, a > 0$ and set $\lambda_n = \lambda_{p,\tau}\sigma := (\sqrt{1+a} + \tau^{-1})\sigma\sqrt{2\log p/n}$ in (1.3). Then if $\beta$ is $s$-sparse with $\delta_{2s} + \theta_{s,2s} < 1 - \tau$, the Dantzig selector obeys with probability at least $1 - (\sqrt{\pi \log p}p^a)^{-1}$, $\left\|\widehat{\beta} - \beta\right\|_2^2 \leq 2C_2^2(\sqrt{1+a} + \tau^{-1})^2 \log p \left(\sigma^2/n + \sum_{i=1}^p \min\left(\beta_i^2, \sigma^2/n\right)\right).$*

From this point on we let $\delta := \delta_{2s}$ and $\theta := \theta_{s,2s}$; Analysis in [5] (Theorem 2) and the current paper yields the following constants, where $C_3$ has not been optimized,

$$C_2 = 2C_0' + \frac{1+\delta}{1-\delta-\theta} \quad \text{where } C_0' = \frac{C_0}{1-\delta-\theta} + \frac{\theta(1+\delta)}{(1-\delta-\theta)^2}, \qquad (3.2)$$

where $C_0 = 2\sqrt{2}\left(1 + \frac{1-\delta^2}{1-\delta-\theta}\right) + (1 + 1/\sqrt{2})\frac{(1+\delta)^2}{1-\delta-\theta}$; We now define

$$C_1 = C_0' + \frac{1+\delta}{1-\delta-\theta} \quad \text{and} \quad C_3^2 = 3(\sqrt{1+a} + \tau^{-1})^2((C_0' + C_4)^2 + 1) + \frac{4(1+a)}{\Lambda_{\min}^2(2s_0)}. \qquad (3.3)$$

We first set up the notation following that in [5]. We order the $\beta_j$'s in decreasing order of magnitude

$$|\beta_1| \geq |\beta_2| \ldots \geq |\beta_p|. \qquad (3.4)$$

Recall that $s_0$ is the smallest integer such that $\sum_{i=1}^p \min(\beta_i^2, \lambda^2\sigma^2) \leq s_0\lambda^2\sigma^2$, where $\lambda = \sqrt{2\log p/n}$. Thus by definition of $s_0$, as essentially shown in [5], that $0 \leq s_0 \leq s$ and

$$s_0\lambda^2\sigma^2 \leq \lambda^2\sigma^2 + \sum_{i=1}^p \min(\beta_i^2, \lambda^2\sigma^2) \leq 2\log p\left(\frac{\sigma^2}{n} + \sum_{i=1}^p \min\left(\beta_i^2, \frac{\sigma^2}{n}\right)\right) \quad (3.5)$$

$$\text{and} \quad s_0\lambda^2\sigma^2 \geq \sum_{j=1}^{s_0+1} \min(\beta_j^2, \lambda^2\sigma^2) \geq (s_0+1)\min(\beta_{s_0+1}^2, \lambda^2\sigma^2) \text{ for } s < p, \qquad (3.6)$$

which implies that $\min(\beta_{s_0+1}^2, \lambda^2\sigma^2) < \lambda^2\sigma^2$ and hence by (3.4),

$$|\beta_j| < \lambda\sigma \quad \text{for all } j > s_0. \tag{3.7}$$

We now show in Lemma 3.2 that thresholding at the level of $C\lambda\sigma$ at step 1 selects a set $I$ of at most $2s_0$ variables, among which at most $s_0$ are from $S^c$.

**Lemma 3.2** *Choose $\tau > 0$ such that $\delta_{2s} + \theta_{s,2s} < 1 - \tau$. Let $\beta_{\text{init}}$ be the $\ell_1$-minimizer subject to the constraints, for $\lambda := \sqrt{2\log p/n}$ and $\lambda_{p,\tau} := (\sqrt{1+a} + t^{-1})\sqrt{2\log p/n}$,*

$$\left\|\frac{1}{n}X^T(Y - X\beta_{\text{init}})\right\|_\infty \leq \lambda_{p,\tau}\sigma. \tag{3.8}$$

*Given some constant $C_4 \geq C_1$, for $C_1$ as in (3.3), choose a thresholding parameter $t_0$ so that*

$$C_4\lambda_{p,\tau}\sigma \geq t_0 > C_1\lambda_{p,\tau}\sigma; \quad \text{Set } I = \{j : |\beta_{j,\text{init}}| > t_0\}.$$

*Then with probability at least $\mathbb{P}(\mathcal{T}_a)$, as detailed in Proposition 3.1, we have for $C_0'$ as in (3.2),*

$$\begin{aligned}
|I| &\leq 2s_0, \text{ and } |I \cup S| \leq s + s_0, \text{ and} & (3.9)\\
\|\beta_{\mathcal{D}}\|_2 &\leq \sqrt{(C_0' + C_4)^2 + 1}\lambda_{p,\tau}\sigma\sqrt{s_0}, \text{ where } \mathcal{D} := \{1,\ldots,p\} \setminus I. & (3.10)
\end{aligned}$$

Next we show that even if we miss some columns of $X$ in $S$, we can still hope to get the convergence rate as required in Theorem 1.2 so long as $\|\beta_{\mathcal{D}}\|_2$ is bounded and $I$ is sufficiently sparse, for example, as bounded in Lemma 3.2. We first show in Lemma 3.3 a general result on rate of convergence of the OLS estimator based on a chosen model $I$, where a subset of relevant variables are missing.

**Lemma 3.3** (**OLS estimator with missing variables**) *Let $\mathcal{D} := \{1,\ldots,p\} \setminus I$ and $S_R = \mathcal{D} \cap S$ such that $I \cap S_R = \emptyset$. Suppose $|I \cup S_R| \leq 2s$. Then we have on $\mathcal{T}_a$, for the least squares estimator based on $I$, $\widehat{\beta}_I = (X_I^T X_I)^{-1} X_I^T Y$, it holds that*

$$\left\|\widehat{\beta}_I - \beta\right\|_2^2 \leq \left(\left(\theta_{|I|,|S_R|}\|\beta_{\mathcal{D}}\|_2 + \lambda_{\sigma,a,p}\sqrt{|I|}\right)/\Lambda_{\min}(|I|)\right)^2 + \|\beta_{\mathcal{D}}\|_2^2.$$

Now Theorem 1.2 is an immediate corollary of Lemma 3.2 and 3.3 in view of (3.5), given that $|S_R| < s$, and $|I| \leq 2s_0$ and $|I \cup S_R| \leq |I \cup S| \leq s + s_0 \leq 2s$ as in Lemma 3.2 (3.9). Hence it is clear by (3.10) that we cannot cut too many "significant" variables; in particular, for those that are larger $\lambda\sigma\sqrt{s_0}$, we can cut at most a constant number of them.

## 4 Linear sparsity and random matrices

A special case of design matrices that satisfy the Restricted Eigenvalue assumptions are the random design matrices. This is shown in a large body of work, for example [3, 4, 5, 1, 13], which shows that the uniform uncertainty principle (UUP) holds for "generic" or random design matrices for very significant values of $s$. For example, it is well known that for a random matrix with i.i.d. Gaussian variables (that is, Gaussian Ensemble, subject to normalizations of columns), and the Bernoulli and Subgaussian Ensembles [1, 13], the UUP holds for $s = O(n/\log(p/n))$; hence the thresholding procedure can recover a sparse model using nearly a constant number of measurements per non-zero component despite the stochastic noise, when $n$ is a nonnegligible fraction of $p$. See [5] for other examples of random designs. In our simulations as shown in Section 5, exact recovery rate of the sparsity pattern is very high for a few types of random matrices using a two-step procedure, once the number of samples passes a certain threshold. For example, for an i.i.d. Gaussian Ensemble, the threshold for exact recovery is $n = \Theta(s\log(p/n))$, where $\Theta$ hides a very small constant, when $\beta_{\min}$ is sufficiently large; this shows a strong contrast with the ordinary Lasso, for which the probability of success in terms of exact recovery of the sparsity pattern tends to zero when $n < 2s\log(p - s)$ [19]. In an ongoing work, the author is exploring thresholding algorithms for a broader class of random designs that satisfy the Restricted Eigenvalue assumptions.

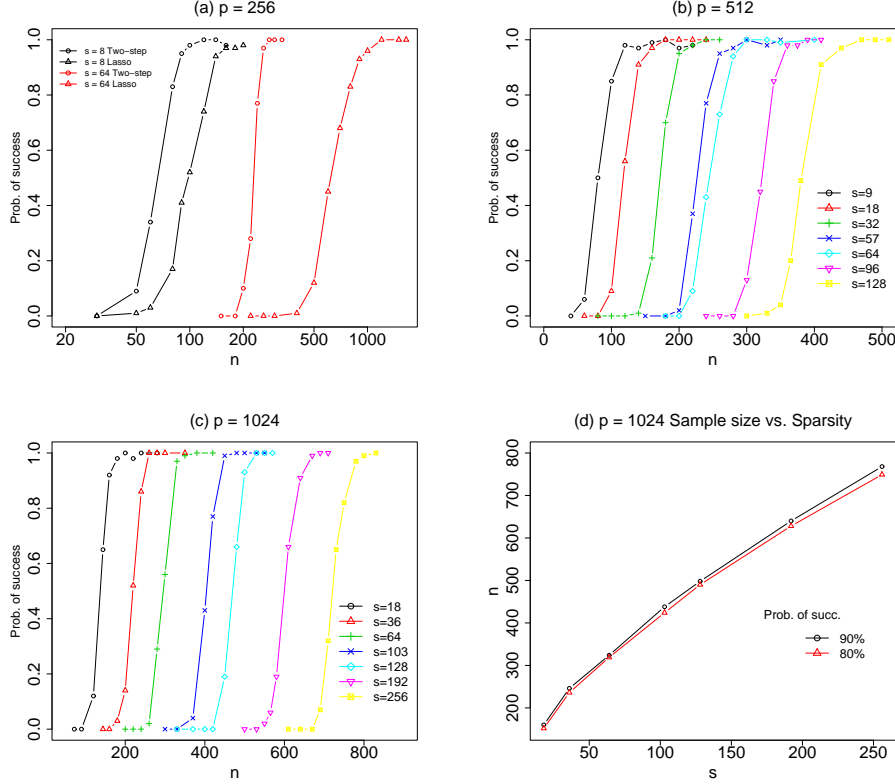

Figure 1: (a) Compare the probability of success under $s = 8$ and $64$ for $p = 256$. The two-step procedure requires much fewer samples than the ordinary Lasso. (b) (c) show the probability of success of the two-step procedure under different levels of sparsity when $n$ increases for $p = 512$ and $1024$ respectively; (d) The number of samples $n$ increases almost linearly with $s$ for $p = 1024$.

## 5  Illustrative experiments

In our implementation, we choose to use the Lasso as the initial estimator. We show in Figure 1 that the two-step procedure indeed recovers a sparse model using a small number of samples per non-zero component in $\beta$ when $X$ is a Gaussian Ensemble. Similar behavior was also observed for the Bernoulli Ensemble in our simulations. We run under three cases of $p = 256, 512, 1024$; for each $p$, we increase the sparsity $s$ by roughly equal steps from $s = 0.2p/\log 0.2p$ to $p/4$. For each tuple $(p, s, n)$, we first generate a random Gaussian Ensemble of size $n \times p$ as $X$, where $X_{ij} \sim N(0, 1)$, which is then normalized to have column $\ell_2$-norm $\sqrt{n}$. For a given $(p, s, n)$ and $X$, we repeat the following experiment 100 times: **1)** Generate a vector $\beta$ of length $p$: within $\beta$ randomly choose $s$ non-zero positions; for each position, we assign a value of $0.9$ or $-0.9$ randomly. **2)** Generate a vector $\epsilon$ of length $p$ according to $N(0, I_p)$, where $I_p$ is the identity matrix. **3)** Compute $Y = X\beta + \epsilon$. $Y$ and $X$ are then fed to the two-step procedure to obtain $\widehat{\beta}$. **4)** We then compare $\widehat{\beta}$ with $\beta$; if all components match in signs, we count this experiment as a success. At the end of the 100 experiments, we compute the percentage of successful runs as the probability of success. We compare with the ordinary Lasso, for which we search over the full path of LARS [8] and always choose the $\widehat{\beta}$ that best matches $\beta$ in terms of support. Inside the two-step procedure, we always fix $\lambda_n \approx 0.69\sqrt{2\log p/n}$ and threshold $\beta_{\text{init}}$ at $t_0 = f_t\sqrt{\frac{\log p}{n}}\sqrt{\widehat{s}}$, where $\widehat{s} = |\widehat{S}_0|$ for $\widehat{S}_0 = \{j : \beta_{j,\text{init}} \geq 0.5\lambda_n\}$, and $f_t$ is a constant chosen from the range of $[1/6, 1/3]$.

**Acknowledgments.**  This research was supported by the Swiss National Science Foundation (SNF) Grant 20PA21-120050/1. The author thanks Larry Wasserman, Sara van de Geer and Peter Bühlmann for helpful discussions, comments and their kind support throughout this work.

# References

[1] R. G. Baraniuk, M. Davenport, R. A. DeVore, and M. B. Wakin. A simple proof of the restricted isometry property for random matrices. *Constructive Approximation*, 28(3):253–263, 2008.

[2] P. J. Bickel, Y. Ritov, and A. B. Tsybakov. Simultaneous analysis of Lasso and Dantzig selector. *The Annals of Statistics*, 37(4):1705–1732, 2009.

[3] E. Candès, J. Romberg, and T. Tao. Stable signal recovery from incomplete and inaccurate measurements. *Communications in Pure and Applied Mathematics*, 59(8):1207–1223, August 2006.

[4] E. Candès and T. Tao. Decoding by Linear Programming. *IEEE Trans. Info. Theory*, 51:4203–4215, 2005.

[5] E. Candès and T. Tao. The Dantzig selector: statistical estimation when p is much larger than n. *Annals of Statistics*, 35(6):2313–2351, 2007.

[6] S. S. Chen, D. L. Donoho, and M. A. Saunders. Atomic decomposition by basis pursuit. *SIAM Journal on Scientific and Statistical Computing*, 20:33–61, 1998.

[7] D. L. Donoho and I. M. Johnstone. Ideal spatial adaptation by wavelet shrinkage. *Biometrika*, 81:425–455, 1994.

[8] B. Efron, T. Hastie, I. Johnstone, and R. Tibshirani. Least angle regression. *Annals of Statistics*, 32(2):407–499, 2004.

[9] E. Greenshtein and Y. Ritov. Persistency in high dimensional linear predictor-selection and the virtue of over-parametrization. *Bernoulli*, 10:971–988, 2004.

[10] V. Koltchinskii. Dantzig selector and sparsity oracle inequalities. *Bernoulli*, 15(3):799–828, 2009.

[11] N. Meinshausen and P. Bühlmann. High dimensional graphs and variable selection with the Lasso. *Annals of Statistics*, 34(3):1436–1462, 2006.

[12] N. Meinshausen and B. Yu. Lasso-type recovery of sparse representations for high-dimensional data. *Annals of Statistics*, 37(1):246–270, 2009.

[13] S. Mendelson, A. Pajor, and N. Tomczak-Jaegermann. Uniform uncertainty principle for bernoulli and subgaussian ensembles. *Constructive Approximation*, 28(3):277–289, 2008.

[14] D. Needell and J. A. Tropp. CoSaMP: Iterative signal recovery from incomplete and inaccurate samples. *Applied and Computational Harmonic Analysis*, 26(3):301–321, 2008.

[15] D. Needell and R. Vershynin. Signal recovery from incomplete and inaccurate measurements via regularized orthogonal matching pursuit. *IEEE Journal of Selected Topics in Signal Processing, to appear*, 2009.

[16] R. Tibshirani. Regression shrinkage and selection via the Lasso. *J. Roy. Statist. Soc. Ser. B*, 58(1):267–288, 1996.

[17] S. A. van de Geer. The deterministic Lasso. *The JSM Proceedings, American Statistical Association*, 2007.

[18] S. A. van de Geer. High-dimensional generalized linear models and the Lasso. *The Annals of Statistics*, 36:614–645, 2008.

[19] M. Wainwright. Sharp thresholds for high-dimensional and noisy sparsity recovery using $\ell_1$-constrained quadratic programming. *IEEE Trans. Inform. Theory*, 2008. to appear, also posted as Technical Report 709, 2006, Department of Statistics, UC Berkeley.

[20] L. Wasserman and K. Roeder. High dimensional variable selection. *The Annals of Statistics*, 37(5A):2178–2201, 2009.

[21] P. Zhao and B. Yu. On model selection consistency of Lasso. *Journal of Machine Learning Research*, 7:2541–2567, 2006.

[22] S. Zhou, S. van de Geer, and P. Bühlmann. Adaptive Lasso for high dimensional regression and gaussian graphical modeling, 2009. arXiv:0903.2515.

[23] H. Zou. The adaptive Lasso and its oracle properties. *Journal of the American Statistical Association*, 101:1418–1429, 2006.

